# Stochastic Learning Networks and their Electronic Implementation

Joshua Alspector*, Robert B. Allen, Victor Hu†, and Srinagesh Satyanarayana‡
Bell Communications Research, Morristown, NJ 07960

We describe a family of learning algorithms that operate on a recurrent, symmetrically connected, neuromorphic network that, like the Boltzmann machine, settles in the presence of noise. These networks learn by modifying synaptic connection strengths on the basis of correlations seen locally by each synapse. We describe a version of the supervised learning algorithm for a network with analog activation functions. We also demonstrate unsupervised competitive learning with this approach, where weight saturation and decay play an important role, and describe preliminary experiments in reinforcement learning, where noise is used in the search procedure. We identify the above described phenomena as elements that can unify learning techniques at a physical microscopic level.

These algorithms were chosen for ease of implementation in vlsi. We have designed a CMOS test chip in 2 micron rules that can speed up the learning about a millionfold over an equivalent simulation on a VAX 11/780. The speedup is due to parallel analog computation for summing and multiplying weights and activations, and the use of physical processes for generating random noise. The components of the test chip are a noise amplifier, a neuron amplifier, and a 300 transistor adaptive synapse, each of which is separately testable. These components are also integrated into a 6 neuron and 15 synapse network. Finally, we point out techniques for reducing the area of the electronic correlational synapse both in technology and design and show how the algorithms we study can be implemented naturally in electronic systems.

## 1. INTRODUCTION

There has been significant progress, in recent years, in modeling brain function as the collective behavior of highly interconnected networks of simple model neurons. This paper focuses on the issue of learning in these networks especially with regard to their implementation in an electronic system. Learning phenomena that have been studied include associative memory[1], supervised learning by error correction[2] and by stochastic search[3], competitive learning[4] [5] reinforcement learning[6], and other forms of unsupervised learning[7]. From the point of view of neural plausibility as well as electronic implementation, we particularly like learning algorithms that change synaptic connection strengths asynchronously and are based only on information available locally at the synapse. This is illustrated in Fig. 1, where a model synapse uses only the correlations of the neurons it connects and perhaps some weak global evaluation signal not specific to individual neurons to decide how to adjust its conductance.

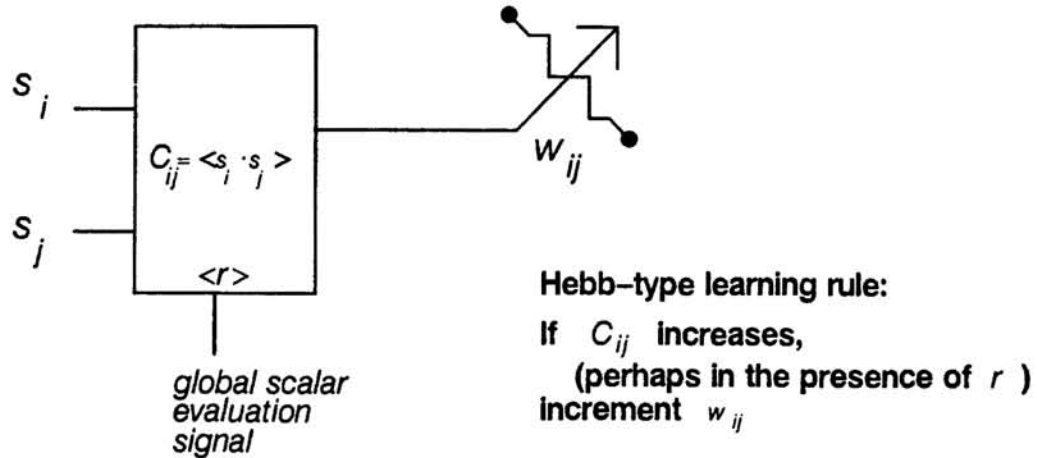

Hebb-type learning rule:

**If** $C_{ij}$ **increases,**
(perhaps in the presence of $r$ )
**increment** $w_{ij}$

Fig. 1. A local correlational synapse.

We believe that a stochastic search procedure is most compatible with this viewpoint. Statistical procedures based on noise form the communication pathways by which global optimization can take place based only on the interaction of neurons. Search is a necessary part of any learning procedure as the network attempts to find a connection strength matrix that solves a particular problem. Some learning procedures attack the search directly by gradient following through error correction[8] [9] but electronic implementation requires specifying which neurons are input, hidden and output in advance and necessitates global control of the error correction[2] procedure in a way that requires specific connectivity and synchrony at the neural level. There is also the question of how such procedures would work with unsupervised methods and whether they might get stuck in local minima. Stochastic processes can also do gradient following but they are better at avoiding minima, are compatible with asynchronous updates and local weight adjustments, and, as we show in this paper, can generalize well to less supervised learning.

The phenomena we studied are 1) analog activation, 2) noise, 3) semi-local Hebbian synaptic modification, and 4) weight decay and saturation. These techniques were applied to problems in supervised, unsupervised, and reinforcement learning. The goal of the study was to see if these diverse learning styles can be unified at the microscopic level with a small set of physically plausible and electronically implementable phenomena. The hope is to point the way for powerful electronic learning systems in the future by elucidating the conditions and the types of circuits that may be necessary. It may also be true that the conditions for electronic learning may

have some bearing on the general principles of biological learning.

## 2. LOCAL LEARNING AND STOCHASTIC SEARCH

### 2.1 Supervised Learning in Recurrent Networks with Analog Activations

We have previously shown[10] how the supervised learning procedure of the Boltzmann machine[3] can be implemented in an electronic system. This system works on a recurrent, symmetrically connected network which can be characterized as settling to a minimum in its Liapunov function[1][11]. While this architecture may stretch our criterion of neural plausibility, it does provide for stability and analyzability. The feedback connectivity provides a way for a supervised learning procedure to propagate information back through the network as the stochastic search proceeds. More plausible would be a randomly connected network where symmetry is a statistical approximation and inhibition damps oscillations, but symmetry is more efficient and well matched to our choice of learning rule and search procedure.

We have extended our electronic model of the Boltzmann machine to include analog activations. Fig. 2 shows the model of the neuron we used and its *tanh* or *sigmoid* transfer function. The net input consists of the usual weighted sum of activations from other neurons but, in the case of Boltzmann machine learning, these are added to a noise signal chosen from a variety of distributions so that the neuron performs the physical computation:

$$activation = f(net_i) = f(\Sigma w_{ij} s_j + noise) = \tanh(gain * net_i)$$

Instead of counting the number of on-on and off-off cooccurrences of neurons which a synapse connects, the correlation rule now defines the value of a cooccurrence as:

$$C_{ij} = f_i * f_j$$

where $f_i$ is the activation of neuron $i$ which is a real value from -1 to 1. Note that this rule effectively counts both on-on and off-off cooccurrences in the high gain limit. In this limit, for Gaussian noise, the cumulative probability distribution for the neuron to have activation +1 (on) is close to sigmoidal. The effect of noise "jitter" is illustrated at the bottom of the figure. The weight change rule is still:

$$\text{if } C_{ij}{}^+ > C_{ij}{}^- \text{ then increment } w_{ij} \dots \text{ else decrement}$$

where the plus phase clamps the output neurons in their desired states while the minus phase allows them to run free.

As mentioned, we have studied a variety of noise distributions other than those based on the Boltzmann distribution. The 2-2-1 XOR problem was selected as a test case since it has been shown[10] to be easily caught in local minima. The gain was manipulated in conditions with no noise or with noise sampled from one of three distributions. The Gaussian distribution is closest to true electronic thermal noise such as used in our implementation, but we also considered a cut-off uniform distribution and a Cauchy distribution with long noise tails for comparison. The inset to Fig. 3 shows a histogram of samples from the noise distributions used. The noise was multiplied by the temperature to 'jitter' the transfer function. Hence, the jitter decreased as the annealing schedule proceeded.

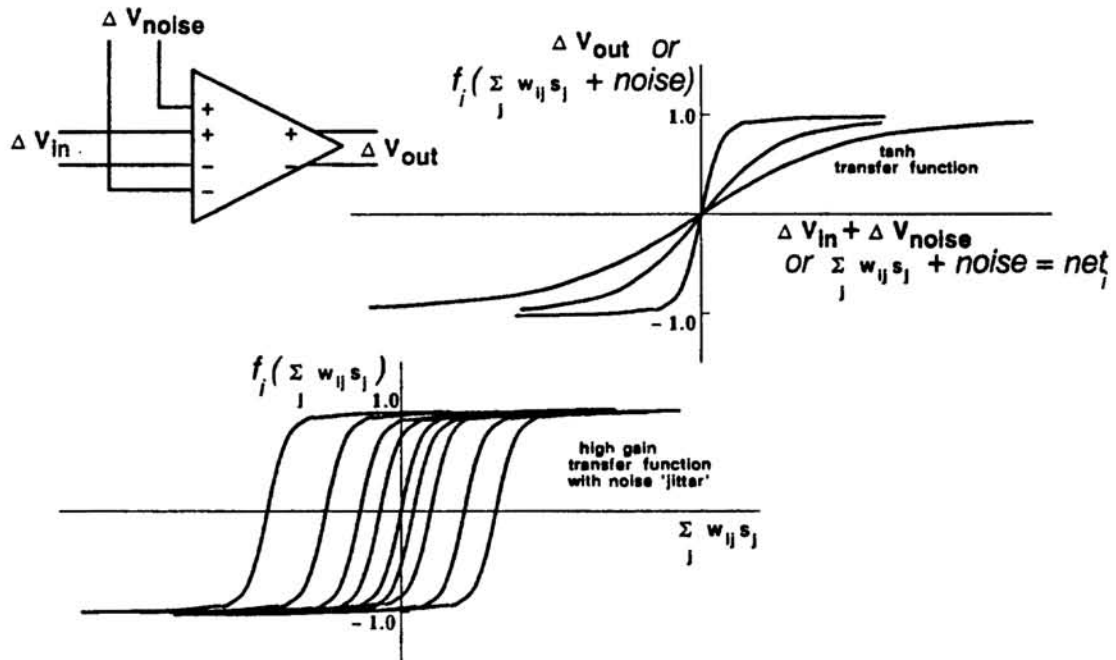

Fig. 2. Electronic analog neuron.

Fig. 3 shows average performance across 100 runs for the last 100 patterns of 2000 training pattern presentations. It can be seen that reducing the gain from a sharp step can improve learning in a small region of gain, even without noise. There seems to be an optimal gain level. However, the addition of noise for any distribution can substantially improve learning at all levels of gain.

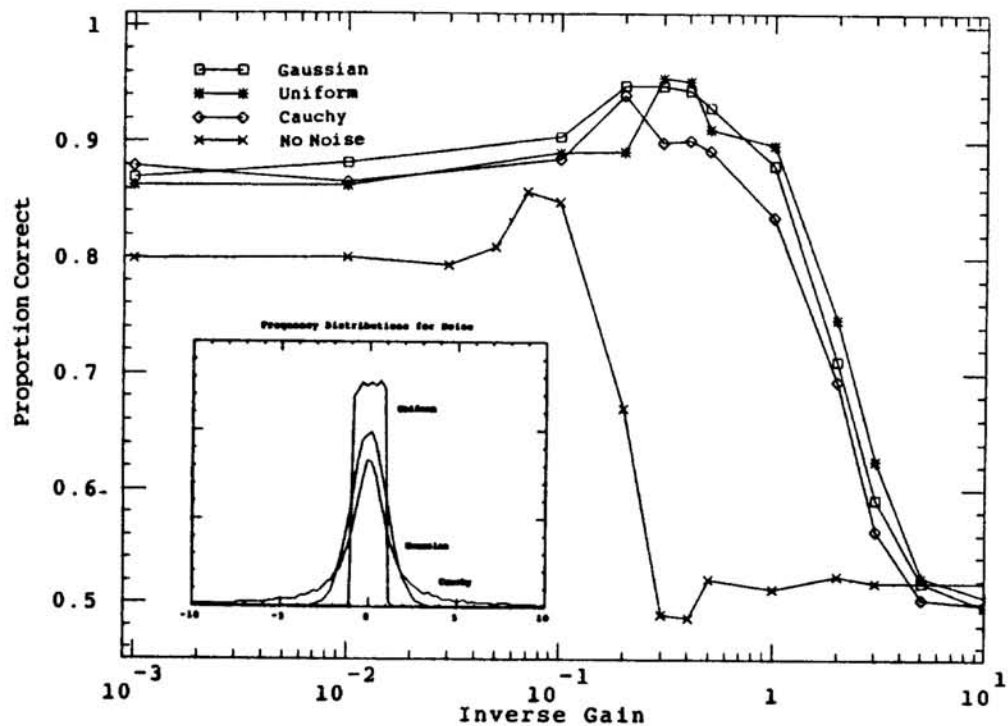

Fig. 3. Proportion correct vs. inverse gain.

## 2.2 Stochastic Competitive Learning

We have studied how competitive learning[4][5] can be accomplished with stochastic local units. After the presentation of the input pattern, the network is annealed and the weight is increased between the winning cluster unit and the input units which are on. As shown in Fig. 4 this approach was applied to the dipole problem of Rumelhart and Zipser. A 4×4 pixel array input layer connects to a 2 unit competitive layer with recurrent inhibitory connections that are not adjusted. The inhibitory connections provide the competition by means of a winner-take-all process as the network settles. The input patterns are dipoles — only two input units are turned on at each pattern presentation and they must be physically adjacent, either vertically or horizontally. In this way, the network learns about the connectedness of the space and eventually divides it into two equal spatial regions with each of the cluster units responding only to dipoles from one of the halves. Rumelhart and Zipser renormalized the weights after each pattern and picked the winning unit as the one with the highest activation. Instead of explicit normalization of the weights, we include a decay term proportional to the weight. The weights between the input layer and cluster layer are incremented for on-on correlations, but here there are no alternating phases so that even this gross synchrony is not necessary. Indeed, if small time constants are introduced to the weight updates, no external timing should be needed.

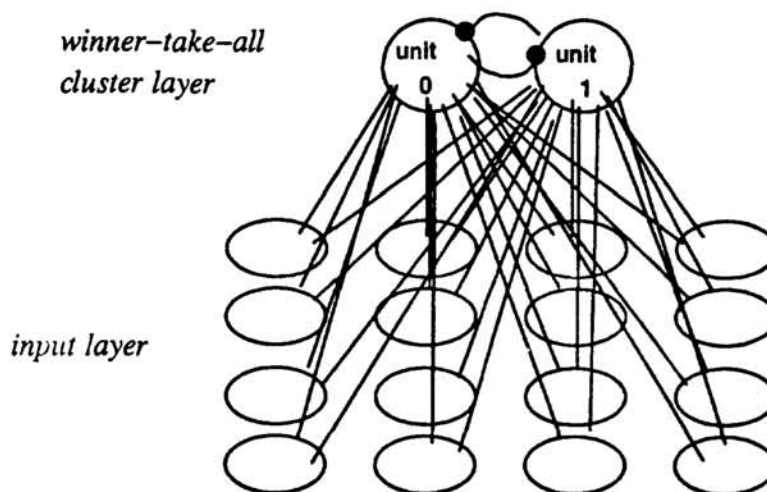

Fig. 4. Competitive learning network for the dipole problem.

Fig. 5 shows the results of several runs. A 1 at the position of an input unit means that unit 1 of the cluster layer has the larger weight leading to it from that position. A + between two units means the dipole from these two units excites unit 1. A 0 and − means that unit 0 is the winner in the complementary case. Note that adjacent 1's should always have a + between them since both weights to unit 1 are stronger. If, however, there is a 1 next to a 0, then there is a tension in the dipole and a competition for dominance in the cluster layer. We define a figure of merit called "surface tension" which is the number of such dipoles in dispute. The smaller the number, the

better. Note in Runs A and B, the number is reduced to 4, the minimum possible value, after 2000 pattern presentations. The space is divided vertically and horizontally, respectively. Run C has adopted a less favorable diagonal division with a surface tension of 6.

```
              Number of dipole pattern presentations

          0         200        800       1400       2000

      0-0-0-0    1+0-0+1    1+1+1+1    1+1+1+1    1+1+1+1
      - - - -    + + + +    + + + -    + + + +    + + + +
      0-0-0-0    1+1+1+1    1+1+1-0    1+1+1+1    1+1+1+1
Run A - - - -    + + - -    + + - -    + - + -    - - - +
      0-0-0-0    1+1-0-0    1-0-0-0    0-0-0-0    0-0-0-0
      - - - -    + - - -    - - - -    - - - -    - - - -
      0-0-0-0    0-0-0-0    0-0-0-0    0-0-0-0    0-0-0-0

      0-0-0-0    0-0-0-0    0-0-0+1    0-0-0-1    0-0-1+1
      - - - -    - - - +    - - + +    - - - +    - - + +
      0-0-0-0    0-0-0+1    0-0-1+1    0-0-1+1    0-0-1+1
Run B - - - -    - - - +    - - + +    - - + +    - - + +
      0-0-0-0    1-0-1+1    0-0-1+1    0-0-1+1    0-0-1+1
      - - - -    + - + +    - - + +    - - + +    - - + +
      0-0-0-0    1+0+1+1    0-0+1+1    0-0+1+1    0-0+1+1

      0-0-0-0    0+1+1+1    0+1+1+1    1+1+1+1    1+1+1+1
      - - - -    - + + +    - + + +    + + + +    - + + +
      0-0-0-0    0-1+1+1    0+1+1+1    0+1+1+1    0-0+1+1
Run C - - - -    - + + +    - + + +    - - + +    - - + +
      0-0-0-0    0-1+1+1    0-0-0-0    0-0-0-0    0-0-0-1
      - - - -    - - - -    - - - -    - - - -    - - - +
      0-0-0-0    0-0-0-0    0-0-0-0    0-0-0-0    0-0-0-1
```

Fig. 5. Results of competitive learning runs on the dipole problem.

Table 1 shows the result of several competitive algorithms compared when averaged over 100 such runs. The deterministic algorithm of Rumelhart and Zipser gives an average surface tension of 4.6 while the stochastic procedure is almost as good. Note that noise is essential in helping the competitive layer settle. Without noise the surface tension is 9.8, showing that the winner-take-all procedure is not working properly.

| Competitive learning algorithm | "surface tension" |
|---|---|
| **Stochastic net with decay** | |
| **– anneal: T=30→ T=1.0** | **4.8** |
| **– no anneal: 70 @ T=1.0** | **9.8** |
| **Stochastic net with renormalization** | **5.6** |
| **Deterministic, winner–take–all (Rumelhart & Zipser)** | **4.6** |

Table 1. Performance of competitive learning algorithms across 100 runs.

We also tried a procedure where, instead of decay, weights were renormalized. The model is that each neuron can support a maximum amount of weight leading into it. Biologically, this might be the area that other neurons can form synapses on, so that one synapse cannot increase its strength except at the expense of some of the others. Electronically, this can be implemented as

current emanating from a fixed current source per neuron. As shown in Table 1, this works nearly as well as decay. Moreover, preliminary results show that renormalization is especially effective when more then two cluster units are employed.

Both of the stochastic algorithms, which can be implemented in an electronic synapse in nearly the same way as the supervised learning algorithm, divide the space just as the deterministic normalization procedure[4] does. This suggests that our chip can do both styles of learning, supervised if one includes both phases and unsupervised if only the procedure of the minus phase is used.

### 2.3 Reinforcement Learning

We have tried several approaches to reinforcement learning using the synaptic model of Fig. 1 where the evaluation signal is a scalar value available globally that represents how well the system performed on each trial. We applied this model to an xor problem with only one output unit. The reinforcement was $r = 1$ for the correct output and $r = -1$ otherwise. To the network, this was similar to supervised learning since for a single unit, the output state is fully specified by a scalar value. A major difference, however, is that we do not clamp the output unit in the desired state in order to compare plus and minus phases. This feature of supervised learning has the effect of adjusting weights to follow a gradient to the desired state. In the reinforcement learning described here, there is no plus phase. This has a satisfying aspect in that no overall synchrony is necessary to compare phases, but is also much slower at converging to a solution because the network has to search the solution space without the guidance of a teacher clamping the output units. This situation becomes much worse when there is more than one output unit. In that case, the probability of reinforcement goes down exponentially with the number of outputs. To test multiple outputs, we chose the simple replication problem whereby the output simply has to replicate the input. We chose the number of hidden units equal to the input (or output).

In the absence of a teacher to clamp the outputs, the network has to find the answer by chance, guided only by a "critic" which rates its effort as "better" or "worse". This means the units must somehow search the space. We use the same stochastic units as in the supervised or unsupervised techniques, but now it is important to have the noise or the annealing temperature set to a proper level. If it is too high, the reinforcement received is random rather than directed by the weights in the network. If it is too low, the available states searched become too small and the probability of finding the right solution decreases. We tuned our annealing schedule by looking at a volatility measure defined at each neuron which is simply the fraction of the time the neuron activation is above zero. We then adjust the final anneal temperature so that this number is neither 0 or 1 (noise too low) nor 0.5 (noise too high). We used both a fixed annealing schedule for all neurons and a unit-specific schedule where the noise was proportional to the sum of weight magnitudes into the unit. A characteristic of reinforcement learning is that the percent correct initially increases but then decreases and often oscillates widely. To avoid this, we added a factor of $(1 - <r>)$ multiplying the final temperature. This helped to stabilize the learning.

In keeping with our simple model of the synapse, we chose a weight adjustment technique that consisted of correlating the states of the connected neurons with the global reinforcement signal. Each synapse measured the quantity $R = rs_i s_j$ for each pattern presented. If $R > 0$, then $w_{ij}$ is incremented and it is decremented if $R < 0$. We later refined this procedure by insisting that the reinforcement be greater than a recent average so that $R = (r - <r>)s_i s_j$. This type of procedure

appears in previous work in a number of forms.[12] [13] For $r = \pm 1$ only, this "excess reinforcement" is the same as our previous algorithm but differs if we make a comparison between short term and long term averages or use a graded reinforcement such as the negative of the sum squared error. Following a suggestion by G. Hinton, we also investigated a more complex technique whereby each synapse must store a time average of three quantities: $<r>$, $<s_i s_j>$, and $<rs_i s_j>$. The definition now is $R = <rs_i s_j> - <r><s_i s_j>$ and the rule is the same as before. Statistically, this is the same as "excess reinforcement" if the latter is averaged over trials. For the results reported below the values were collected across 10 pattern presentations. A variation, which employed a continuous moving average, gave similar results.

Table 2 summarizes the performance on the xor and the replication task of these reinforcement learning techniques. As the table shows a variety of increasingly sophisticated weight adjustment rules were explored; nevertheless we were unable to obtain good results with the techniques described for more than 5 output units. In the third column, a small threshold had to be exceeded prior to weight adjustment. In the fourth column, unit-specific temperatures dependent on the sum of weights, were employed. The last column in the table refers to frequency dependent learning where we trained on a single pattern until the network produced a correct answer and then moved on to another pattern. This final procedure is one of several possible techniques related to 'shaping' in operant learning theory in which difficult patterns are presented more often to the network.

| network | t=1 | time-averaged | + ε=0.1 | +T-ΣW | +freq |
|---------|-----|---------------|---------|-------|-------|
| xor | | | | | |
| 2-4-1 | (0.60) 0.64 | (0.70) 0.88 | (0.76) 0.88 | (0.92) 0.99 | (0.98) 1.00 |
| 2-2-1 | (0.58) 0.57 | (0.69) 0.74 | (0.96) 1.00 | (0.85) 1.00 | (0.78) 0.88 |
| eplication | | | | | |
| 2-2-2 | (0.94) 0.94 | (0.46) 0.46 | (0.91) 0.97 | (0.87) 0.99 | (0.97) 1.00 |
| 3-3-3 | (0.15) 0.21 | (0.31) 0.33 | (0.31) 0.62 | (0.37) 0.37 | (0.97) 1.00 |
| 4-4-4 | - | - | - | - | (0.75) 1.00 |
| 5-5-5 | - | - | - | - | (0.13) 0.87 |
| 6-6-6 | - | - | - | - | (0.02) 0.03 |

Table 2. Proportion correct performance of reinforcement learning
after (2K) and 10K patterns.

Our experiments, while incomplete, hint that reinforcement learning can also be implemented by the same type of local-global synapse that characterize the other learning paradigms. Noise is also necessary here for the random search procedure.

**2.4 Summary of Study of Fundamental Learning Parameters**

In summary, we see that the use of noise and our model of a local correlational synapse with a non-specific global evaluation signal are two important features in all the learning paradigms. Graded activation is somewhat less important. Weight decay seems to be quite important although saturation can substitute for it in unsupervised learning. Most interesting from our point of view is that all these phenomena are electronically implementable and therefore physically

plausible. Hopefully this means they are also related to true neural phenomena and therefore provide a basis for unifying the various approaches of learning at a microscopic level.

## 3. ELECTRONIC IMPLEMENTATION

### 3.1 The Supervised Learning Chip

We have completed the design of the chip previously proposed.[10] Its physical style of computation speeds up learning a millionfold over a computer simulation. Fig. 6 shows a block diagram of the neuron. It is a double differential amplifier. One branch forms a sum of the inputs from the differential outputs of all other neurons with connections to it. The other adds noise from the noise amplifier. This first stage has low gain to preserve dynamic range at the summing nodes. The second stage has high gain and converts to a single ended output. This is fed to a switching arrangement whereby either this output state or some externally applied desired state is fed into the final set of inverter stages which provide for more gain and guaranteed digital complementarity.

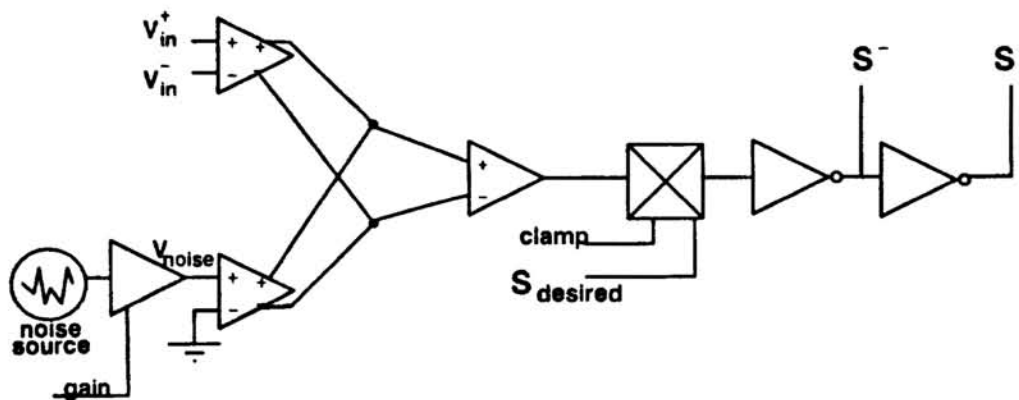

Fig. 6. Block diagram of neuron.

The noise amplifier is shown schematically in Fig. 7. Thermal noise, with an rms level of tens of microvolts, from the channel of an FET is fed into a 3 stage amplifier. Each stage provides a potential gain of 100 over the noise bandwidth. Low pass feedback in each stage stabilizes the DC output as well as controls gain and bandwidth by means of an externally controlled variable resistance for tuning the annealing cycle.

Fig. 8 shows a block diagram of the synapse. The weight is stored in 5 flip-flops as a sign and magnitude binary number. These flip-flops control the conductance from the outputs of neuron $i$ to the inputs of neuron $j$ and vice-versa as shown in the figure. The conductance of the FETs are in the ratio 1:2:4:8 to correspond to the value of the binary number while the sign bit determines whether the true or complementary lines connect. The flip-flops are arranged in a counter which is controlled by the correlation logic. If the plus phase correlations are greater than the minus phase, then the counter is incremented by a single unit. If less, it is decremented.

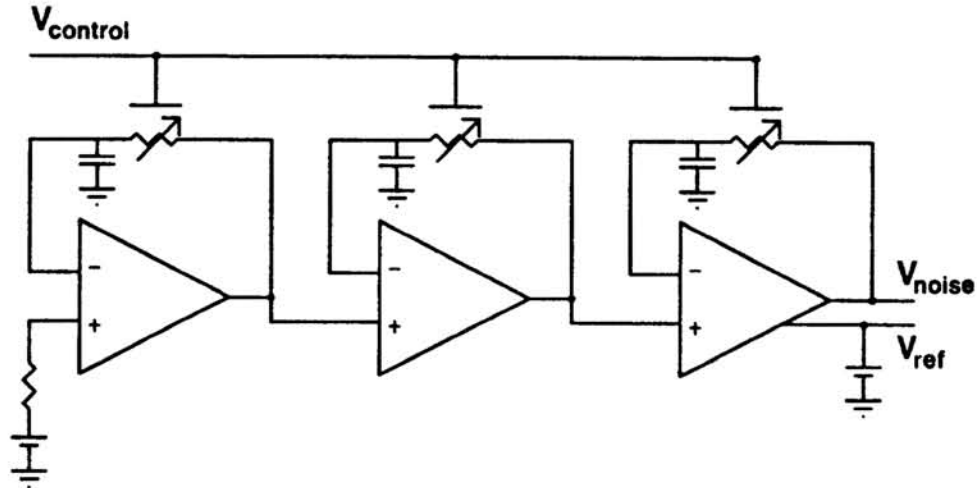

Fig. 7. Block diagram of noise amplifier.

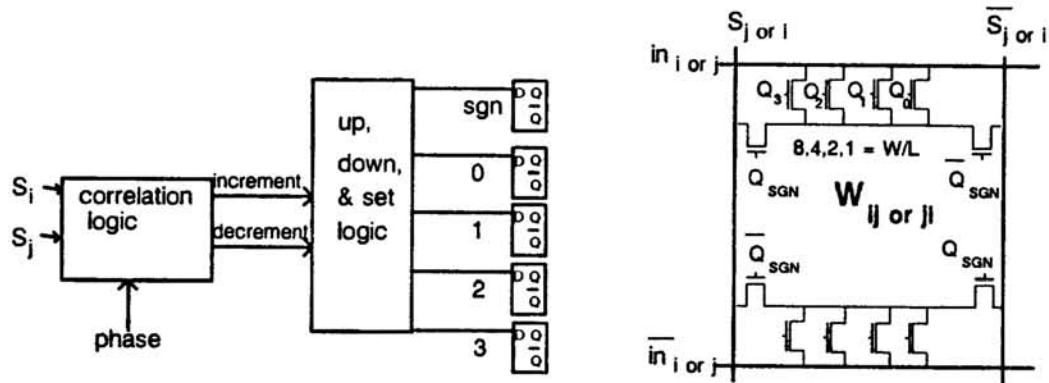

Fig. 8. Block diagram of synapse.

Fig. 9 shows the layout of a test chip. A 6 neuron, 15 synapse network may be seen in the lower left corner. Each neuron has attached to it a noise amplifier to assure that the noise is uncorrelated. The network occupies an area about 2.5 mm on a side in 2 micron design rules. Each 300 transistor synapse occupies 400 by 600 microns. In contrast, a biological synapse occupies only about one square micron. The real miracle of biological learning is in the synapse where plasticity operates on a molecular level, not in the neuron. We can't hope to compete using transistors, however small, especially in the digital domain. Aside from this small network, the rest of the chip is occupied with test structures of the various components.

### 3.2 Analog Synapse

Analog circuit techniques can reduce the size of the synapse and increase its functionality. Several recent papers[14] [15] have shown how to make a voltage controlled resistor in MOS technology. The voltage controlling the conductance representing the synaptic weight can be obtained by an analog charge integrator from the correlated activation of the neurons which the synapse in question connects. A charge integrator with a "leaky capacitor" has a time constant

which can be used to make comparisons as a continuous time average over the last several trials, thereby adding temporal information. One can envision this time constant as being adaptive as well. The charge integrator directly implements the analog Hebb-type[16] correlation rules of section 2.

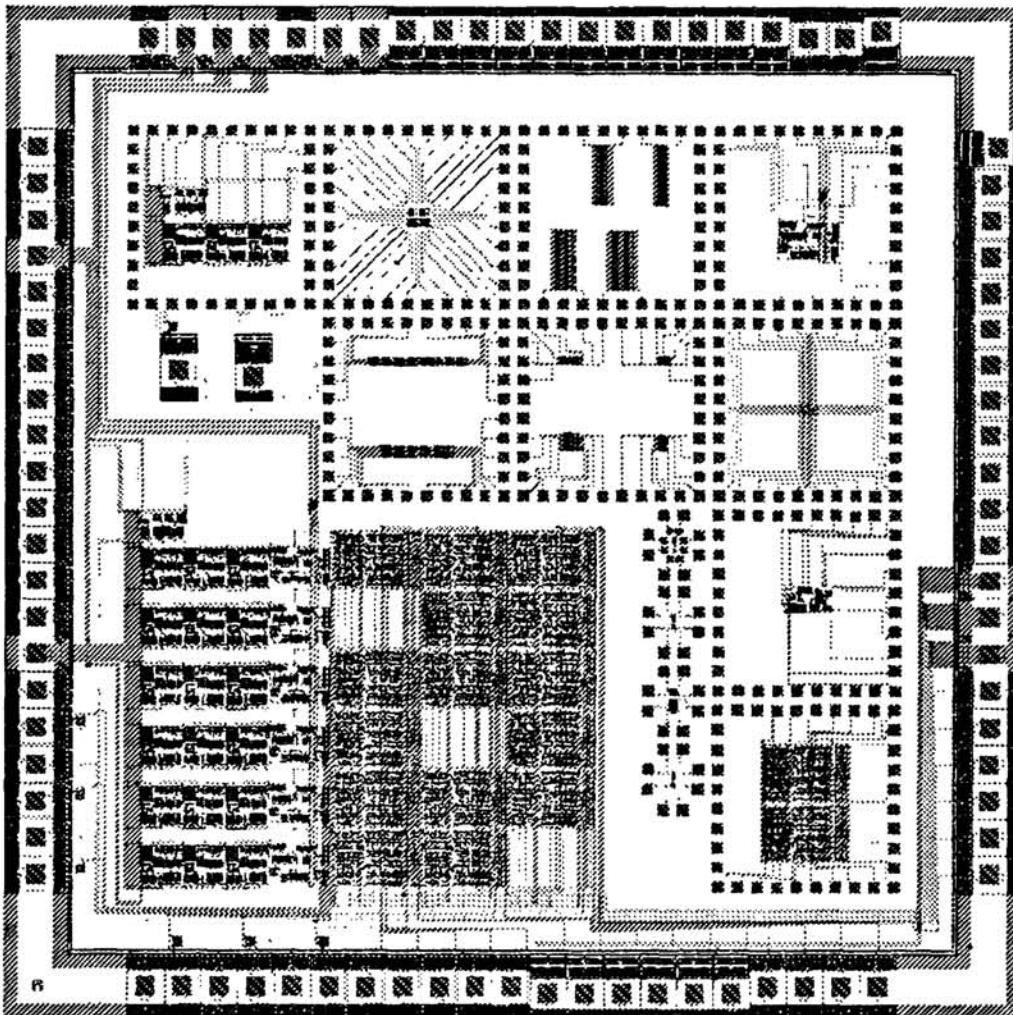

Fig. 9. Chip layout.

### 3.3 Technological Improvements for Electronic Neural Networks

It is still necessary to store the voltage which controls the analog conductance and we propose the EPROM[17] or EEPROM device for this. Such a device can hold the value of the weight in the same way that flip-flops do in the digital implementation of the synapse[10]. The process which creates this device has two polysilicon layers which are useful for making high valued capacitances in analog circuitry. In addition, the second polysilicon layer could be used to make CCD devices for charge storage and transport. Coupled with the charge storage on a floating gate[18], this forms a compact, low power representation for weight values that approach biological values. Another useful addition would be a high valued stable resistive layer[19]. One

could thereby avoid space-wasting long-channel MOSFETs which are currently the only reasonable way to achieve high resistance in MOS technology. Lastly, the addition of a diffusion step or two creates a Bi-CMOS process which adds high quality bipolar transistors useful in analog design. Furthermore, one gets the logarithmic dependence of voltage on current in bipolar technology in a natural, robust way, that is not subject to the variations inherent in using MOSFETs in the subthreshold region. This is especially useful in compressing the dynamic range in sensory processing[20].

## 4. CONCLUSION

We have shown how a simple adaptive synapse which measures correlations can account for a variety of learning styles in stochastic networks. By embellishing the standard CMOS process and using analog design techniques, a technology suitable for implementing such a synapse electronically can be developed. Noise is an important element in our formulation of learning. It can help a network settle, interpolate between discrete values of conductance during learning, and search a large solution space. Weight decay ("forgetting") and saturation are also important for stability. These phenomena not only unify diverse learning styles but are electronically implementable.

**ACKNOWLEDGMENT:**
This work has been influenced by many researchers. We would especially like to thank Andy Barto and Geoffrey Hinton for valuable discussions on reinforcement learning, Yannis Tsividis for contributing many ideas in analog circuit design, and Joel Gannett for timely releases of his vlsi verification software.

## Footnotes

* Address for correspondence: J. Alspector, Bell Communications Research, 2E-378, 435 South St., Morristown, NJ 07960 / (201) 829-4342 / josh@bellcore.com

† Permanent address: University of California, Berkeley, EE Department, Cory Hall, Berkeley, CA 94720

‡ Permanent address: Columbia University, EE Department, S.W. Mudd Bldg., New York, NY 10027

## References

1. J.J. Hopfield, "Neural networks and physical systems with emergent collective computational abilities", Proc. Natl. Acad. Sci. USA **79**, 2554-2558 (1982).

2. D.E. Rumelhart, G.E. Hinton, and R.J. Williams, "Learning internal representations by error propagation", in *Parallel Distributed Processing: Explorations in the Microstructure of Cognition. Vol. 1: Foundations*, edited by D.E. Rumelhart and J.L. McClelland, (MIT Press, Cambridge, MA, 1986), p. 318.

3. D.H. Ackley, G.E. Hinton, and T.J. Sejnowski, "A learning algorithm for Boltzmann machines", Cognitive Science **9**, 147-169 (1985).

4. D.E. Rumelhart and D. Zipser, "Feature discovery by competitive learning", Cognitive Science **9**, 75-112 (1985).

5. S. Grossberg, "Adaptive pattern classification and universal recoding: Part I. Parallel development and coding of neural feature detectors.", Biological Cybernetics **23**, 121-134 (1976).

6. A.G. Barto, R.S. Sutton, and C.W. Anderson, "Neuronlike adaptive elements that can solve difficult learning control problems", IEEE Trans. Sys. Man Cyber. **13**, 835 (1983).

7. B.A. Pearlmutter and G.E. Hinton, "G-Maximization: An unsupervised learning procedure for discovering regularities", in *Neural Networks for Computing*, edited by J.S. Denker, AIP Conference Proceedings 151, American Inst. of Physics, New York (1986), p.333.

8. F. Rosenblatt, *Principles of Neurodynamics: Perceptrons and the Theory of Brain Mechanisms* (Spartan Books, Washington, D.C., 1961).

9. G. Widrow and M.E. Hoff, "Adaptive switching circuits", Inst. of Radio Engineers, Western Electric Show and Convention, Convention Record, Part 4, 96-104 (1960).

10. J. Alspector and R.B. Allen, "A neuromorphic vlsi learning system", in *Advanced Research in VLSI: Proceedings of the 1987 Stanford Conference.* edited by P. Losleben (MIT Press, Cambridge, MA, 1987), pp. 313-349.

11. M.A. Cohen and S. Grossberg, "Absolute stability of global pattern formation and parallel memory storage by competitive neural networks", Trans. IEEE **13**, 815, (1983).

12. B. Widrow, N.K. Gupta, and S. Maitra, "Punish/Reward: Learning with a critic in adaptive threshold systems", IEEE Trans. on Sys. Man & Cyber., SMC-3, 455 (1973).

13. R.S. Sutton, "Temporal credit assignment in reinforcement learning", unpublished doctoral dissertation, U. Mass. Amherst, technical report COINS 84-02 (1984).

14. Z. Czarnul, "Design of voltage-controlled linear transconductance elements with a matched pair of FET transistors", IEEE Trans. Circ. Sys. **33**, 1012, (1986).

15. M. Banu and Y. Tsividis, "Floating voltage-controlled resistors in CMOS technology", Electron. Lett. **18**, 678-679 (1982).

16. D.O. Hebb, *The Organization of Behavior* (Wiley, NY, 1949).

17. D. Frohman-Bentchkowsky, "FAMOS - a new semiconductor charge storage device", Solid-State Electronics **17**, 517 (1974).

18. J.P. Sage, K. Thompson, and R.S. Withers, "An artificial neural network integrated circuit based on MNOS/CCD principles", in *Neural Networks for Computing*, edited by J.S. Denker, AIP Conference Proceedings 151, American Inst. of Physics, New York (1986), p.381.

19. A.P. Thakoor, J.L. Lamb, A. Moopenn, and J. Lambe, "Binary synaptic connections based on memory switching in a-Si:H", in *Neural Networks for Computing*, edited by J.S. Denker, AIP Conference Proceedings 151, American Inst. of Physics, New York (1986), p.426.

20. M.A. Sivilotti, M.A. Mahowald, and C.A. Mead, "Real-Time visual computations using analog CMOS processing arrays", in *Advanced Research in VLSI: Proceedings of the 1987 Stanford Conference.* edited by P. Losleben (MIT Press, Cambridge, MA, 1987), pp. 295-312.
